# Optimal cue selection strategy

**Vidhya Navalpakkam**
Department of Computer Science
USC, Los Angeles
navalpak@usc.edu

**Laurent Itti**
Department of Computer Science
USC, Los Angeles
itti@usc.edu

## Abstract

Survival in the natural world demands the selection of relevant visual cues to rapidly and reliably guide attention towards prey and predators in cluttered environments. We investigate whether our visual system selects cues that guide search in an optimal manner. We formally obtain the optimal cue selection strategy by maximizing the signal to noise ratio ($\mathcal{SNR}$) between a search target and surrounding distractors. This optimal strategy successfully accounts for several phenomena in visual search behavior, including the effect of target-distractor discriminability, uncertainty in target's features, distractor heterogeneity, and linear separability. Furthermore, the theory generates a new prediction, which we verify through psychophysical experiments with human subjects. Our results provide direct experimental evidence that humans select visual cues so as to maximize $\mathcal{SNR}$ between the targets and surrounding clutter.

## 1   Introduction

Detecting a yellow tiger among distracting foliage in different shades of yellow and brown requires efficient top-down strategies that select relevant visual cues to enable rapid and reliable detection of the target among several distractors. For simple scenarios such as searching for a red target, the Guided Search theory [17] predicts that search efficiency can be improved by boosting the red feature in a top-down manner. But for more complex and natural scenarios such as detecting a tiger in the jungle or looking for a face in a crowd, finding the optimum amount of top-down enhancement to be applied to each low-level feature dimension encoded by the early visual system is non-trivial. It must not only consider features present in the target, but also those present in the distractors. In this paper, we formally obtain the optimal cue selection strategy and investigate whether our visual system has evolved to deploy it. In section 2, we formulate cue selection as an optimization problem where the relevant goal is to maximize the signal to noise ratio ($\mathcal{SNR}$) of the saliency map, so that the target becomes most salient and quickly draws attention, thereby minimizing search time. Next, we show through simulations that this optimal top-down guided search theory successfully accounts for several observed phenomena in visual search behavior, such as the effect of target-distractor discriminability, uncertainty in target's features, distractor heterogeneity, linear separability, and more. In section 4, we describe the design and analysis of psychophysics experiments to test new, counter-intuitive predictions of the theory. The results of our study suggest that humans deploy optimal cue selection strategies to detect targets in cluttered and distracting environments.

## 2 Formalizing visual search as an optimization problem

To quickly find a target among distractors, we wish to maximize the salience of the target relative to the distractors. Thus we can define the signal to noise ratio ($\mathcal{SNR}$) as the ratio of salience of the target to the distractors. Assuming that visual cues or features are encoded by populations of neurons in early visual areas, we define the optimal cue selection strategy as the best choice of neural response gain that maximizes the signal to noise ratio ($\mathcal{SNR}$). In the rest of this section, we formally obtain the optimal choice of gain in neural responses that will maximize $\mathcal{SNR}$.

$\mathcal{SNR}$ **in a visual search paradigm:** In a typical visual search paradigm, the salience of the target and distractors is a random variable that depends on their location in the search array, their features, the spatial configuration of target and distractors, and that varies between identical repeated trials due to internal noise in neural response to the visual input. Hence, we express $\mathcal{SNR}$ as the ratio of expected salience of the target over expected salience of the distractors, with the expectation taken over all possible target and distractor locations, their features and spatial configurations, and over several repeated trials.

$$\mathcal{SNR} = \frac{\text{Mean salience of the Target}}{\text{Mean salience of the distractor}}$$

**Search array and its stimuli:** Let search array $A$ be a two-dimensional display that consists of one target $T$ and several distractors $D_j$ ($j = 1...N^2$-1). Let the display be divided into an invisible $N \times N$ grid, with one item occuring at each cell $(x, y)$ in the grid. Let the color, contrast, orientation and other target parameters $\theta_T$ be chosen from a distribution $P(\theta|T)$. Similarly, for each distractor $D_j$, let its parameters $\theta_{D_j}$ be sampled independently from a distribution $P(\theta|D)$. Thus, search array $A$ has a fixed choice of target and distractor parameters. Next, the spatial configuration $C$ is decided by a random permutation of some assignment of the target and distractors to the $N^2$ cells in $A$ (such that there is exactly one item in each cell). Thus, for a given search array $A$, the spatial configuration as well as stimulus parameters are fixed. Finally, given a choice of parameter $\theta$ and its spatial location $(x, y)$, we generate an image pattern $R(\theta)$ (a set of pixels and their values) and embed it at location $(x, y)$ in search array $A$. Thus, we generate search array $A$.

**Saliency computation:** Let the input search array $A$ be processed by a population of neurons with gaussian tuning curves tuned to different stimulus parameters such as $\mu_1, \mu_2, ...\mu_n$. The output of this early visual processing stage is used to compute saliency maps $s_i(x, y, A)$ of search array $A$, that consist of the visual salience at every location $(x, y)$ for feature-values $\mu_i(i = 1...n)$. Let $s_i(x, y, A)$ be combined linearly to form $S(x, y, A)$, the overall salience at location $(x, y)$. Further, assuming a multiplicative gain $g_i$ on the $i^{th}$ saliency map, we obtain:

$$S(x, y, A) \quad = \quad \sum_i g_i s_i(x, y, A) \tag{1}$$

**Salience of the target and distractors:** Let $S_T(A)$ be a random variable representing the salience of the target $T$ in search array $A$. To factor out the variability due to internal noise $\eta$, we consider $E_\eta[S_T(A)]$, which is the mean salience of the target over repeated identical presentations of $A$. Further, let $E_C[S_T(A)]$ be the mean salience of the target averaged over all spatial configurations of a given set of target and distractor parameters. Similarly, $E_{\theta|T}[S_T(A)]$ is the mean salience of the target over all target parameters. The mean salience of the target combined over several repeated presentations of the search array $A$ (to factor out internal noise $\eta$), over all spatial configurations $C$, and over all choices of

target parameters $\theta|T$ is given below. Further, since $\eta$, $C$ and $\theta$ are independent random variables, we can rewrite the joint expectation as follows:

$$E[S_T(A)] \quad = \quad E_{\theta|T}[E_C[E_\eta[S_T(A)]]] \tag{2}$$

Let $S_D(A)$ represent the mean salience of distractors $D_j$ ($j = 1...N^2$-1) in search array $A$. Similar to computing the mean salience of the target, we find the mean salience of distractors over all $\eta$, $C$ and $\theta|D$.

$$S_D(A) \quad = \quad E_{D_j}[s_{iD_j}(A)] \tag{3}$$

$$E[S_D(A)] \quad = \quad E_{\theta|D}[E_C[E_\eta[S_D(A)]]] \tag{4}$$

$\mathcal{SNR}$ **and its optimization:** The additive salience and multiplicative gain hypothesis in eqn. 1 yields the following:

$$E[S_T(A)] \quad = \quad \sum_{i=1}^{n} g_i E_{\Theta|T}[E_C[E_\eta[s_{iT}(A)]]] \tag{5}$$

$$E[S_D(A)] \quad = \quad \sum_{i=1}^{n} g_i E_{\Theta|T}[E_C[E_\eta[s_{iT}(A)]]] \text{ (similarly)} \tag{6}$$

$\mathcal{SNR}$ can be expressed in terms of salience as:

$$\mathcal{SNR} \quad = \quad \frac{\sum_{i=1}^{n} g_i E_{\Theta|T}[E_C[E_\eta[s_{iT}(A)]]]}{\sum_{i=1}^{n} g_i E_{\Theta|D}[E_C[E_\eta[s_{iD}(A)]]]} \tag{7}$$

We wish to find the optimal choice of $g_i$ that maximises $\mathcal{SNR}$. Hence, we differentiate $\mathcal{SNR}$ wrt $g_i$ to get the following:

$$\frac{\partial}{\partial g_i}\mathcal{SNR} \quad = \quad \frac{\frac{E_{\Theta|T}[E_C[E_\eta[s_{iT}(A)]]]}{E_{\Theta|D}[E_C[E_\eta[s_{iD}(A)]]]} - \frac{\sum_{j=1}^{n} g_j E_{\Theta|T}[E_C[E_\eta[s_{jT}(A)]]]}{\sum_{j=1}^{n} g_j E_{\Theta|D}[E_C[E_\eta[s_{jD}(A)]]]}}{\frac{\sum_{j=1}^{n} g_j E_{\Theta|D}[E_C[E_\eta[s_{jD}(A)]]]}{E_{\Theta|D}[E_C[E_\eta[s_{iD}(A)]]]}} \tag{8}$$

$$= \quad \frac{\frac{\mathcal{SNR}_i}{\mathcal{SNR}} - 1}{\alpha_i} \tag{9}$$

where $\alpha_i$ is a normalization term and $\mathcal{SNR}_i$ is the signal-to-noise ratio of the $i^{th}$ saliency map.

$$\mathcal{SNR}_i = E_{\Theta|T}[E_C[E_\eta[s_{iT}(A)]]]/E_{\Theta|D}[E_C[E_\eta[s_{iD}(A)]]] \tag{10}$$

The sign of the derivative, $\left(\frac{d}{dg_i}\mathcal{SNR}\right)_{g_i=1}$ tells us whether $g_i$ should be increased, decreased or maintained at the baseline activation 1 in order to maximize $\mathcal{SNR}$.

$$\frac{\mathcal{SNR}_i}{\mathcal{SNR}} \quad < \quad 1 \Rightarrow \frac{d}{dg_i}\mathcal{SNR} < 0 \Rightarrow \mathcal{SNR} \text{ increases as } g_i \text{ decreases} \Rightarrow g_i < 1 \tag{11}$$

$$= \quad 1 \Rightarrow \frac{d}{dg_i}\mathcal{SNR} = 0 \Rightarrow \mathcal{SNR} \text{ does not change with } g_i \Rightarrow g_i = 1 \tag{12}$$

$$> \quad 1 \Rightarrow \frac{d}{dg_i}\mathcal{SNR} > 0 \Rightarrow \mathcal{SNR} \text{ increases as } g_i \text{ increases} \Rightarrow g_i > 1 \tag{13}$$

Thus, we obtain an intuitive result that $g_i$ increases as $\frac{\mathcal{SNR}_i}{\mathcal{SNR}}$ increases. We simplify this monotonic relationship assuming proportionality. Further, if we impose a restriction that the gains cannot be increased indiscriminately, but must sum to some constant, say the total number of saliency maps ($n$), we have the following:

$$\text{let } g_i \propto \frac{\mathcal{SNR}_i}{\mathcal{SNR}} \tag{14}$$

$$\text{if } \sum_i g_i = n \quad \Rightarrow \quad g_i = \frac{\mathcal{SNR}_i}{\frac{\sum_i \mathcal{SNR}_i}{n}} \tag{15}$$

Thus the gain of a saliency map tuned to a band of feature-values depends on the strength of the signal-to-noise ratio in that band compared to the mean signal-to-noise ratio in all bands in that feature dimension.

## 3 Predictions of the optimal cue selection strategy

To understand the implications of biasing features according to the optimal cue selection strategy, we simulate a simple model of early visual cortex. We assume that each feature dimension is encoded by a population of neurons with overlapping gaussian tuning curves that are broadly tuned to different features in that dimension. Let $f_i(\theta)$ represent the tuning curve of the $i^{th}$ neuron in a population of broadly tuned neurons with overlapping tuning curves. Let the tuning width $\sigma$ and amplitude $a$ be equal for all neurons, and $\mu_i$ represent the preferred stimulus parameter (or feature) of the $i^{th}$ neuron.

$$f_i(\theta) = \frac{a}{\sigma} \exp\left(-\frac{(\theta - \mu_i)^2}{2\sigma^2}\right) \tag{16}$$

Let $\vec{r}(\Theta(x, y, A)) = \{r_1(\Theta(x, y, A))...r_n(\Theta(x, y, A))\}$ be the population response to a stimulus parameter $\Theta(x, y, A)$ at a location $(x, y)$ in search array $A$, where $r_i$ refers to the response of the $i^{th}$ neuron and $n$ is the total number of neurons in the population. Let the neural response $r_i(\Theta(x, y, A))$ be a Poisson random variable.

$$P(r_i(\Theta(x, y, A)) = z) = P_{f_i(\Theta(x,y,A))}(z) \tag{17}$$

For simplicity, let's assume that the local neural response $r_i(\Theta(x, y, A))$ is a measure of salience $s_i(x, y, A)$. Using eqns. 2, 4, 10, 16, 17, we can derive the mean salience of the target and distractor, and use it to compute $\mathcal{SNR}_i$.

$$s_i(x, y, A) = r_i(\Theta(x, y, A)) \tag{18}$$
$$E[s_{iT}(A)] = E_{\theta|T}[f_i(\theta)] \tag{19}$$
$$E[s_{iD}(A)] = E_{\theta|D}[f_i(\theta)] \tag{20}$$
$$\mathcal{SNR}_i = \frac{E_{\theta|T}[f_i(\theta)]}{E_{\theta|D}[f_i(\theta)]} \tag{21}$$

Finally, the gains $g_i$ on each saliency map can be found using eqn. 15. Thus, for a given distribution of stimulus parameters for the target $P(\theta|T)$ and distractors $P(\theta|D)$, we simulate the above model of early visual cortex, compute salience of target and distractors, compute $\mathcal{SNR}_i$ and obtain $g_i$. In the rest of this section, we plot the distribution of optimal choice of gains $g_i$ for an exhaustive list of conditions where knowledge of the target and distractors varies from complete certainty to uncertainty.

**Unknown target and distractors:** In the trivial case where there is no knowledge of the target and distractors, all cues are equally relevant and the optimal choice of gains is the same as baseline activation (unity). $\mathcal{SNR}$ is minimum leading to a slow search. This prediction is consistent with visual search experiments that observe slow search when the target and distractors are unknown due to reversal between trials [1, 2].

**Search for a known target:** During search for a known target, the optimal strategy predicts that $\mathcal{SNR}$ can be maximised by boosting neurons according to how strongly they respond to the target feature (as shown in figure 1, predicted $\mathcal{SNR}$ is 12.2 dB). Thus, a neuron that is optimally tuned to the target feature receives maximal gain. This prediction is consistent with single unit recordings on feature-based attention which show that the gain in neural response depends on the similarity between the neuron's preferred feature and the target feature [3, 4].

**Role of uncertainty in target features:** When there's uncertainty in the target's features, i.e., when the target's parameter assumes multiple values according to some probability

distribution $P(\theta|T)$, the optimal strategy predicts that $\mathcal{SNR}$ decreases, leading to a slower search (as shown in figure 1, $\mathcal{SNR}$ decreases from 12.2 dB to 9 dB ). This result is consistent with psychophysics experiments which suggest that better knowledge of the target leads to faster search [5, 6].

**Distractor heterogeneity:** While searching for an unknown target among known distractors, the optimal strategy predicts that $\mathcal{SNR}$ can be maximised by suppressing the neurons tuned to the distractors (see figure 1). But as we increase distractor heterogeneity or the number of distractor types, it predicts a decrease in $\mathcal{SNR}$ (from 36 dB to 17 dB, figure 1). This result is consistent with experimental data [10].

**Discriminability between target and distractors:** Several experiments and theories have studied the effect of target-distractor discriminability [10]-[17]. The optimal cue selection strategy also shows that if the target and distractors are very different or highly discriminable, $\mathcal{SNR}$ is high and the search is efficient ($\mathcal{SNR}$ = 51.4 dB, see figure 1). Otherwise, if they are similar and not well separated in feature space, $\mathcal{SNR}$ is low and the search is hard ($\mathcal{SNR}$ = 16.3 dB, see figure 1). Moreover, during search for a less discriminable target from distractors, the optimal strategy predicts that the neuron optimally tuned to the target may not be boosted maximally. Instead, a neuron that is sub-optimally tuned to the target and farther away from the distractors receives maximal gain. This new and counterintuitive prediction is tested by visual search experiments described in the next section.

**Linear separability effect:** The optimal strategy also predicts the linear separability effect [18, 19] which suggests that when the target and distractors are less discriminable, search is easier if the target and distractors can be separated by a line in feature space (see figure 1). This effect has been demonstrated in size (e.g., search for the smallest or largest item is faster than search for a medium-sized item in the display)[20], chromaticity and luminance [21, 19], and orientation [22, 23].

## 4  Testing new predictions of the optimal cue selection strategy

In this section, we describe the design and analysis of psychophysics experiments to verify the counter-intuitive prediction mentioned in the previous section, i.e., during searching for a target that is less discriminable from the distractors, a neuron that is sub-optimally tuned to the target's feature will be boosted more than a neuron that is optimally tuned to the target's feature.

### 4.1  Design of psychophysics experiments

Our experiments are designed in two phases: phase 1 to set up the top-down bias and phase 2 to measure the bias.

**Phase 1 - Setup the top-down bias:** Subjects perform the primary task T1 which is a visual search for the target among distractors. This task sets the top-down bias on cues so that the target becomes the most salient item in the display, thus accelerating target detection. Subjects are trained on T1 trials until their performance stabilises with at least 80% accuracy. They are instructed to find the target ($55°$ tilt) among several distractors ($50°$ tilt). The target and distractors are the same for all T1 trials. To avoid false reports (which may occur due to boredom or lack of attention) and to verify that subjects indeed find the target, we introduce a novel *no cheat* scheme as follows: After finding the target among distractors, subjects press any key. Following the key press, we flash a grid of fineprint random numbers briefly (120ms) and ask subjects to report the number at the target's location. Online feedback on accuracy of report is provided. Thus, the top-down bias is set up by performing T1 trials.

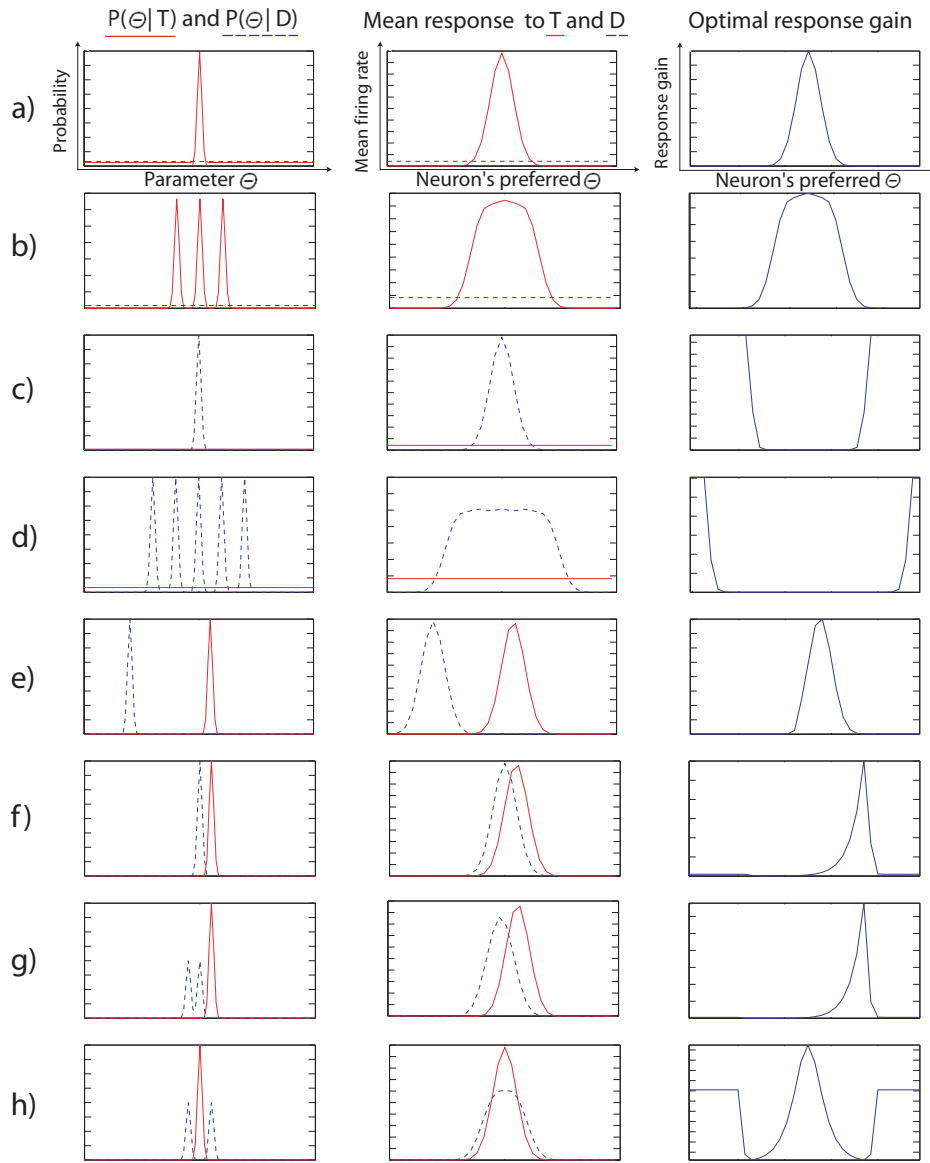

Figure 1: a) Search for a known target – left: Prior knowledge $P(\theta|T)$ has a peak at the known target feature and $P(\theta|D)$ is flat as the distractor is unknown, middle: The expected responses of a population of neurons to the target is highest for neurons tuned around the target's $\theta$ while the expected response to the distractors is flat, right: The optimal response gain in this situation is to boost the gain of the neurons that are tuned around the target's $\theta$; b) Search for an uncertain target; c) Unknown target among a known distractor; d) Presence of heterogeneous distractors; e) High discriminability between target and distractors; f) Low discriminability; g) Search for an extreme feature (linearly separable) among others; h) Search for a mid feature (nonlinearly separable) among others.

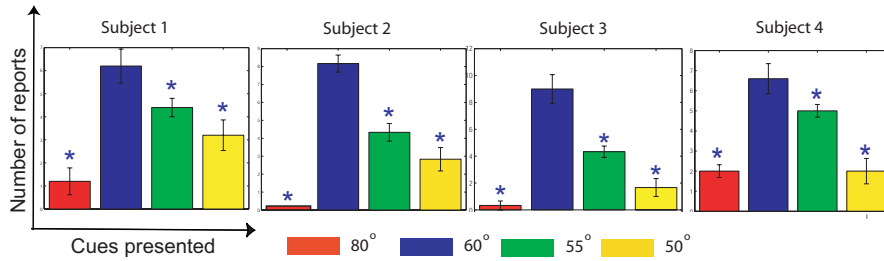

Figure 2: The results of the T2 trials described in section 4.1 (phase 2) are shown here. For each of the four subjects, the number of reports on the steepest ($80°$), relevant ($60°$), target ($55°$) and distractor ($50°$) cues are shown in these bar plots. As predicted by the theory, a paired t-test reveals that the number of reports on the relevant cue is significantly higher ($p < 0.05$) than the number of reports on the target, distractor and steepest cues, as indicated by the blue star.

**Phase 2 - Measure the top-down bias:** To measure the top-down bias generated by the above task, we randomly insert T2 trials in between T1 trials. Our theory predicts that during search for the target ($55°$) among distractors ($50°$), the most relevant cue will be around $60°$ and not $55°$. To test this, we briefly (200ms) flash four cues - steepest (S, $80°$), relevant as predicted by our theory (R, $60°$), target (T, $55°$) and distractor (D, $50°$). A cue that is biased more appears more salient, attracts a saccade, and gets reported. In other words, the greater the top-down bias on a cue, the higher the number of its reports. According to our theory, there should be higher number of reports on R than T.

**Experimental details:** We ran 4 naïve subjects. All were aged 22-30, had normal or corrected vision, volunteered or participated for course credit. As mentioned earlier, each subject received training on T1 trials for a few days until the performance (search speed) stabilised with atleast 80% accuracy. To become familiar with the secondary task, they were trained on 50 T2 trials. Finally, each subject performed 10 blocks of 50 trials each, with T2 trials randomly inserted in between T1 trials.

## 4.2 Results

For each of the four subjects, we extracted the number reports on the steepest ($N_S$), relevant ($N_R$), target ($N_T$) and distractor ($N_D$) cues, for each block. We used a paired t test to check for statistically significant differences between $N_R$ and $N_T$, $N_D$, $N_S$. Results are shown in figure 2. As predicted by the theory, we found a significantly higher number of reports on the relevant cue than the target cue.

## 5 Discussion

In this paper, we have investigated whether our visual system has evolved to use optimal top-down strategies to select relevant cues that quickly and reliably detect the target among distracting environments. We formally obtained the optimal cue selection strategy where cues are chosen such that the signal-to-noise ratio ($\mathcal{SNR}$) of the saliency map is maximized, thus maximizing the target's salience relative to the distractors. The resulting optimal strategy is to boost a cue or feature if it provides higher signal-to-noise ratio than average. Through simulations, we confirmed the predictions of the optimal strategy

with existing experimental data on visual search behavior, including the effect of distractor heterogeneity [10], uncertainty in target's features [5, 6], target-distractor discriminability [10], linear separabilty effect [18, 19]. Our study complements the recent work on optimal eye movement strategies [24]. While we focus on an early stage of visual processing - optimal cue selection in order to create a saliency map with maximum $\mathcal{SNR}$, their study focuses on a later stage of visual processing - optimal saccade generation such that for a given saliency map, the probability of subsequent target detection is maximized. Thus, both optimal cue selection and saccade generation are necessary for optimal visual search.

## Acknowledgements

This work was supported by the National Science Foundation, National Eye Institute, National Imagery and Mapping Agency, Zumberge Innovation Fund, and Charles Lee Powell Foundation.

## References

[1] V Maljkovic and K Nakayama. *Mem Cognit*, 22(6):657–672, Nov 1994.

[2] J. M. Wolfe, S. J. Butcher, and M. Hyle. *J Exp Psychol Hum Percept Perform*, 29(2):483–502, 2003.

[3] S Treue and J C Martinez Trujillo. *Nature*, 399(6736):575–579, Jun 1999.

[4] J. C. Martinez-Trujillo and S. Treue. *Curr Biol*, 14(9):744–751, May 2004.

[5] J. M. Wolfe, T. S. Horowitz, N. Kenner, M. Hyle, and N. Vasan. *Vision Res*, 44(12):1411–1426, Jun 2004.

[6] Timothy J Vickery, Li-Wei King, and Yuhong Jiang. *J Vis*, 5(1):81–92, Feb 2005.

[7] A. Triesman and J. Souther. *Journal of Experimental Psychology: Human Perception and Performance*, 14:107–141, 1986.

[8] A. Treisman and S. Gormican. *Psychological Review 95*, 1:15–48, 1988.

[9] R. Rosenholtz. *Percept Psychophys*, 63(3):476–489, Apr 2001.

[10] J Duncan and G W Humphreys. *Psychological Rev*, 96:433–458, 1989.

[11] A. L. Nagy and R. R. Sanchez. *Journal of the Optical Society of America A 7*, 7:1209–1217, 1990.

[12] H. Pashler. *Percept Psychophys*, 41(4):385–392, Apr 1987.

[13] K. Rayner and D. L. Fisher. *Percept Psychophys*, 42(1):87–100, Jul 1987.

[14] A. Treisman. *J Exp Psychol Hum Percept Perform*, 17(3):652–676, Aug 1991.

[15] J. Palmer, P. Verghese, and M. Pavel. *Vision Res*, 40(10-12):1227–1268, 2000.

[16] J. M. Wolfe, K. R. Cave, and S. L. Franzel. *J. Exper. Psychol.*, 15:419–433, 1989.

[17] J. M. Wolfe. *Psyonomic Bulletin and Review*, 1(2):202–238, 1994.

[18] M. D'Zmura. *Vision Research 31*, 6:951–966, 1991.

[19] B. Bauer, P. Jolicoeur, and W. B. Cowan. *Vision Research 36*, 10:1439–1465, 1996.

[20] A. Treisman and G. Gelade. *Cognitive Psychology*, 12:97–136, 1980.

[21] B. Bauer, P. Jolicoeur, and W. B. Cowan. *Vision Res*, 36(10):1439–1465, May 1996.

[22] J. M. Wolfe, S. R. Friedman-Hill, M. I. Stewart, and K. M. O' Connell. *J Exp Psychol Hum Percept Perform*, 18(1):34–49, Feb 1992.

[23] W. F. Alkhateeb, R. J. Morris, and K. H. Ruddock. *Spat Vis*, 5(2):129–141, 1990.

[24] J. Najemnik, W. S. Geisler. *Nature*, 434(7031):387–391, Mar 2005.
